# The Time-Marginalized Coalescent Prior for Hierarchical Clustering

**Levi Boyles**
Department of Computer Science
University of California, Irvine
Irvine, CA 92617
lboyles@uci.edu

**Max Welling**
Department of Computer Science
University of California, Irvine
Irvine, CA 92617
welling@uci.edu

## Abstract

We introduce a new prior for use in Nonparametric Bayesian Hierarchical Clustering. The prior is constructed by marginalizing out the time information of Kingman's coalescent, providing a prior over tree structures which we call the Time-Marginalized Coalescent (TMC). This allows for models which factorize the tree structure and times, providing two benefits: more flexible priors may be constructed and more efficient Gibbs type inference can be used. We demonstrate this on an example model for density estimation and show the TMC achieves competitive experimental results.

## 1 Introduction

Hierarchical clustering models aim to fit hierarchies to data, and enjoy the property that clusterings of varying size can be obtained by "pruning" the tree at particular levels. In contrast, standard clustering models must specify the number of clusters beforehand, while Nonparametric Bayesian (NPB) clustering models such as the Dirichlet Process Mixture (DPM) [5, 13] directly infer the (effective) number of clusters. Hierarchical clustering is often used in population genetics for inferring ancestral history and bioinformatics for genetic clustering, and has also seen use in computer vision [18, 1] and topic modelling [3, 1].

NPB models are a class of models of growing popularity. Being Bayesian, these models can easily quantify the uncertainty the the resulting inferences, and being nonparametric, they can seamlessly adapt to increasingly complicated data, avoiding the model selection problem. NPB hierarchical clustering models are an important regime of such models, and have been shown to have superior performance to alternative models in many domains [8]. Thus, further advances in the applicability of these models is important.

There has been substantial work on NPB models for hierarchical clustering. Dirichlet Diffusion Trees (DDT) [16], Kingman's Coalescent [9, 10, 4, 20], and Pitman-Yor Diffusion Trees (PYDT) [11] all provide models in which data is generated from a Continuous-Time Markov Chain (CTMC) that lives on a tree that splits (or coalesces) according to some continuous-time process. The nested CRP and DP [3, 17] and Tree-Structured Stick Breaking (TSSB) [1] define priors over tree structures from which data is directly generated.

Although there is extensive and impressive literature on the subject demonstrating its useful clustering properties, NPB hierarchical clustering has yet to see widespred use. The expensive computational cost typically associated with these models is a likely inhibitor to the adoption of these models. The CTMC based models are typically more computationally intensive than the direct generation models, and there has been substantial work in improving the speed of inference in these models. [12] introduces a variational approximation for the DDT, and [7, 6] provide more efficient SMC schemes for the Coalescent. The direct generation models are typically faster, but usually

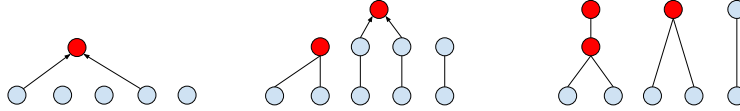

Figure 1: Coalescent tree construction **(left)** A pair is uniformly drawn from $N = 5$ points to coalesce. **(middle)** The coalescence time $t_5$ is drawn from $Exp(\binom{5}{2})$, and another pair on the remaining 4 points is drawn uniformly. **(right)** After drawing $t_4 \sim Exp(\binom{4}{2})$, the coalescence time for the newly coalesced pair is $t_5 + t_4$.

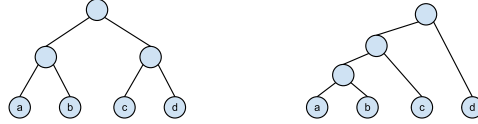

Figure 2: Consider the trees one might construct by uniformly picking pairs of points to join, starting with four leaves $\{a, b, c, d\}$. One can join $a$ and $b$ first, and then $c$ and $d$ (and then the two parents), or $c$ and $d$ and then $a$ and $b$ to construct the tree on the left. By defining a uniform prior over $\phi_n$, and then marginalizing out the order of the internal nodes $\rho$ (equivalently, the order in which pairs are joined), we then have a prior over $\psi_n$ that puts more mass on balanced trees than unbalanced ones. For example the tree on right can only be constructed in one way by node joining.

come at some cost or limitation; for example the TSSB allows (and requires) that data live at some of its internal nodes.

Our contribution is a new prior over tree structures that is simpler than many of the priors described above, yet still retains the exchangeability and consistency properties of a NPB model. The prior is derived by marginalizing out the times and ordering of internal nodes of the coalescent. The remaining distribution is an exchangeable and consistent prior over tree structures. This prior may be used directly with a data generation process, or a notion of time may be reintroduced, providing a prior with a factorization between the tree structures and times. The simplicity of the prior allows for great flexibility and the potential for more efficient inference algorithms. For the purposes of this paper, we focus on one such possible model, wherein we generate branch lengths according to a process similar to stick-breaking.

We introduce the proposed prior on tree structures in Section 2, the distribution over times conditioned on tree structure in 3.1, and the data generation process in 3.2. We show experimental results in Section 4, and conclude in Section 5.

## 2 Coalescent Prior on Tree Structures

### 2.1 Kingman's Coalescent

Kingman's Coalescent provides a prior over balanced, edge-weighted trees, wherein the weights are often interpreted as representing some notion of time. See Figure 1. A coalescent tree can be sampled as follows: start with $n$ points and $n$ dangling edges hanging from them, and all weights set to 0. Sample a time from $Exp(\binom{n}{2})$, and add this value to the weight for each of the dangling edges. Then pick a pair uniformly at random to coalesce (giving rise to their mutual parent, whose new dangling edge has weight 0). Repeat this process on the remaining $n - 1$ points until a full weighted tree is constructed. Note however, that the weights do not influence the choice of tree structures sampled, which suggests we can marginalize out the times and still retain an exchangeable and consistent distribution over trees. What remains is simply a process in which a uniformly chosen pair of points is joined at every iteration.

### 2.2 Coalescent Distribution over Trees

We consider two types of tree-like structures, generic (rooted) unweighted tree graphs which we denote $\psi_n$ living in $\Psi_n$, and trees of the previous type, but with a specified *ordering* $\rho$ on the internal (non-leaf) nodes of the tree, denoted $(\psi_n, \rho) = \phi_n \in \Phi_n$. Marginalizing out the times of the coalescent gives a uniform prior over ordered tree structures $\phi_n$. The order information is

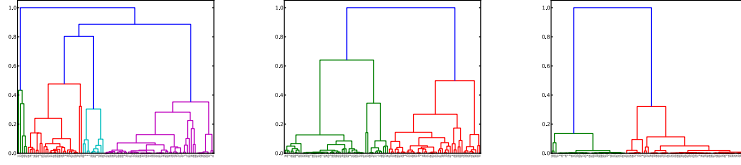

Figure 3: **(left)** A sample from the described prior with stick-breaking parameter $\mathcal{B}(1,1)$ (uniform). **(middle)** A sample using $\mathcal{B}(2,2)$. **(right)** A sample using $\mathcal{B}(4,2)$.

necessary because for a given $\psi_n$ there are multiple ways of constructing it by uniformly picking pairs to join, see Figure 2. If there are $i$ remaining nodes to join, there are $\binom{i}{2}$ ways of joining them, so we have for the probability of a particular $\phi_n$:

$$p(\phi_n) = \prod_{i=2}^{n} \binom{i}{2}^{-1}$$

This defines an exchangeable and consistent prior over $\Phi_n$; exchangeable because $p(\phi_n)$ does not depend on the order in which the data is seen, and consistent because the conditional prior[1] is well defined – we can imagine adding a new leaf to an existing $\phi_n$, which creates a new internal node. Let $y_i$ denote the $i$th internal node[2] of $\phi_n$, $i \in \{1...n-1\}$, and let $y^*$ denote the new internal node. There are $n$ ways of attaching the new internal node below $y_1$, $n-1$ ways of attaching below $y_2$, and so on, giving $\frac{n(n+1)}{2} = \binom{n}{2}$ ways of attaching $y^*$ into $\phi_n$. Thus if we make this choice uniformly at random, we get the probability of the new tree is $p(\phi_{n+1}) = p(\phi_n)\binom{n+1}{2}^{-1} = \prod_{i=2}^{n+1} \binom{i}{2}^{-1}$.

It is possible to marginalize out the ordering information in the coalescent tree structures $\phi_n$ to derive exchangeable, consistent priors on "unordered" tree structures $\psi_n$. We can perform this marginalization by counting how many ordered tree structures $\phi_n \in \Phi_n$ are consistent with a particular unordered tree structure $\psi_n$.

**Lemma 1.** *A tree $\psi_n$ has $T(\psi_n) = \frac{(n-1)!}{\prod_{i=1}^{n-1} m_i}$ possible orderings on its internal nodes, where $m_i$ is the number of internal nodes in the subtree rooted at node $i$.*

(For proof see the supplementary material.) This is in agreement with what we would expect: for an unbalanced tree, $m_i = \{1, 2, ..., n-1\}$, so this gives $T = 1$. Since an unbalanced tree imposes a full ordering on the internal nodes, there can only be one unbalanced ordered tree that maps to the corresponding unbalanced unordered tree. As the tree becomes more balanced, the $m_i$s decrease, increasing $T$.

Thus the probability of a particular $\psi_n$ is $T(\psi_n)$ times the probability of an ordered tree $\phi_n$ under the coalescent:[3]

$$p(\psi_n) = T(\psi_n) \prod_{i=2}^{n} \binom{i}{2}^{-1} = \frac{(n-1)!}{\prod_{i=1}^{n-1} m_i} \prod_{i=2}^{n} \binom{i}{2}^{-1} \tag{1}$$

**Theorem 1.** *$p(\psi_n)$ defines an exchangeable and consistent prior over $\Psi_n$*

$p(\psi_n)$ is clearly still exchangeable as it does not depend on any order of the data, and was defined by marginalizing out a consistent process, so its conditional priors are still well defined and thus $p(\psi_n)$ is consistent. For a more explicit proof see the supplementary material.

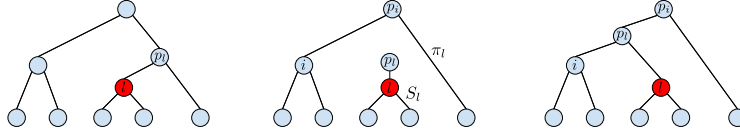

Figure 4: The subtree $S_l$ rooted at the red node $l$ is pruned in preparation for an MCMC move. We perform slice sampling to determine where the pruned subtree's parent should be placed next in the remaining tree $\pi_l$.

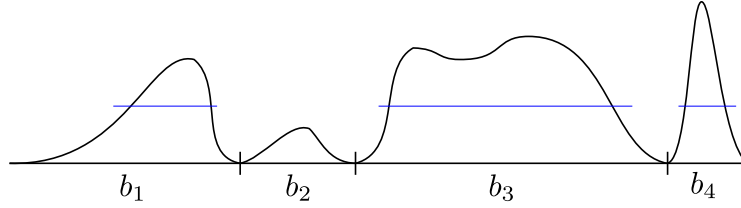

Figure 5: We compute the posterior pdf for each branch that we might attach to. If $\alpha$ or $\beta$ are greater than one, the Beta prior on branch lengths can cause these pdfs to go to zero at the limits of their domain. Thus to enable moves across branches we compute the extrema of each pdf so that all valid intervals are found.

## 3 Data Generation Model

Given a prior over tree structures such as (1), we can define a data generating process in many ways (indeed, any $L_1$ bounded martingale will do [19]); here we restrict our attention to generative models in which we first sample times given a tree structure, and then sample the data according to some process described on those times (in our case Brownian Motion). Examples of other potential data generation models include those in [1], such as the "Generalized Gaussian Diffusions," and the multinomial likelihood often used with the Coalescent.

### 3.1 Branch Lengths

Given a tree structure $\psi$, we can sample branch lengths, $s_i = t_{p_i} - t_i$, with $t_i$ the time of coalescent event $i$, with $t = 0$ at the leaves and $p_i$ is the parent of $i$. Consider the following construction similar to a stick-breaking process: Start with a stick of unit length. Starting at the root, travel down the given $\psi$, and at each split point duplicate the current stick into two sticks, assigning one to each child. Then, sample a Beta random variable $B$ for each of the two sticks where the corresponding children are not leaves. $B$ will be the proportion of the remaining stick attributed to that branch of the tree until the next split point (sticks afterwards will be of length proportional to $(1 - B)$). We have, $B_i = 1 - (t_i/t_{p_i}) = s_i/t_{p_i}$. The total prior over branch lengths can thus be written as:

$$p(\{B_i\}|\psi) = \prod_{i=1}^{N-2} \mathcal{B}(B_i|\alpha, \beta) \tag{2}$$

See Figure 3 for samples from this prior. Note that any distribution whose support is the unit interval may be used, and in fact more innovative schemes for sampling the times may be used as well; one of the major advantages of the TMC over the Coalescent and DDT is that the times may be defined in a variety of ways.

There is a single Beta random variable attributed to each internal node of the tree (except the root, which has $B$ set to 1). Since the order in which we see the data does not affect the way in which we sample these stick proportions, the process remains exchangeable. We denote pairs $(\psi_n, \{B_i\})$ as $\pi$, i.e. a tree structure with branch lengths.

### 3.2 Brownian Motion

Given a tree $\pi$ we can define a likelihood for continuous data $x_i \in \mathbb{R}^p$ using Brownian motion. We denote the length of each branch segment of the tree $s_i$. Data is generated as follows: we start at some unknown location in $\mathbb{R}^p$ at time $t = 1$ and immediately split into two independent Wiener processes (with parameter $\Lambda$), each denoted $y_i$, where $i$ is the index of the relevant branch in $\pi$. Each

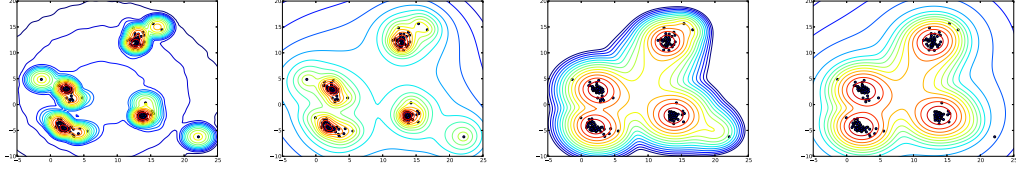

Figure 6: **(left)** Approximated log-density using a DP mixture. **(midleft)** Log-density using a Dirichlet Diffusion Tree model. **(midright)** Log-density using our model directly. **(right)** Log-density using our model with a heavy-tailed noise model at the leaves. Contours are spaced 1 apart, for a total of 15 contours. In the probability domain the various densities look similar.

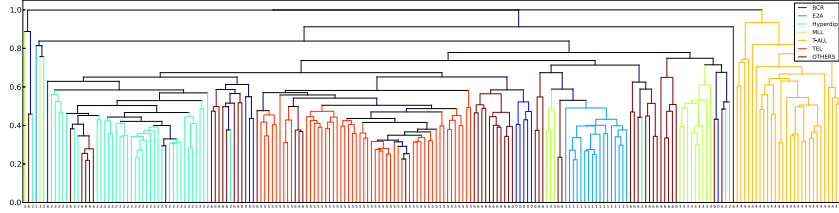

Figure 7: Posterior sample from our model applied to the leukemia dataset. Best viewed in color. Each pure subtree is painted a color unique to the class associated with it. The OTHERS class is a set of datapoints to which no diagnostic label was assigned. A larger view of this figure can be found in the supplementary material.

of the processes evolves for times $s_i$, then, a new independent Wiener process is instantiated at the time of each split, and this continues until all processes reach the leaves of $\pi$ (i.e. $t = 0$), at which point the $y_i$s at the leaves are associated with the data $\mathbf{x}$. This is a similar likelihood to the ones used for Dirichlet Diffusion Trees [16] and the Coalescent [20] for continuous data.

### 3.2.1 Likelihood Computation

The likelihood $p(\mathbf{x}|\pi)$ can be calculated using a single bottom-up sweep of message passing. As in [20], by marginalizing out from the leaves towards the root, the message at an internal node $i$ is a Gaussian with mean $\hat{y}_i$ and variance $\Lambda \nu_i$.

The $\nu$ and $\hat{y}$ messages can be written for any number of incoming nodes in a single form:

$$\nu_i^{-1} = \sum_{j \in c(i)} (\nu_j + s_j)^{-1}; \qquad \hat{y}_i = \nu_i \sum_{j \in c(i)} \frac{\hat{y}_j}{\nu_j + s_j}$$

where $c(i)$ are the nodes sending incoming messages to $i$. We can compute the likelihood using any arbitrary node as the root for message passing. Fixing a particular node as root, we can write the total likelihood of the tree as:

$$p(\mathbf{x}|\pi) = \prod_{i=1}^{n-1} Z_{c(i)}(\mathbf{x}, \pi) \qquad (3)$$

When $|c(i)| = 1$ (e.g. when passing through the root at $t = 1$), $Z_{c(i)} = 1$. When $|c(i)| = 2$ and $|c(i)| = 3$ (when collecting at an arbitrary node $i$ chosen as the root):

$$Z_{l_i, r_i}(\mathbf{x}, \pi) = |2\pi \hat{\Lambda}_i|^{-\frac{1}{2}} \exp\left(-\frac{1}{2}||\hat{y}_{r_i} - \hat{y}_{l_i}||^2_{\hat{\Lambda}_i}\right); \qquad \hat{\Lambda}_i = \Lambda(\nu_{l_i} + \nu_{r_i} + s_{l_i} + s_{r_i}) \qquad (4)$$

$$Z_{p_i, l_i, r_i}(\mathbf{x}, \pi) = |2\pi \Lambda|^{-1} \nu^{*-\frac{k}{2}} e^{\left(-\frac{1}{2}\left(\nu_{p_i}^*||\hat{y}_{l_i} - \hat{y}_{r_i}||^2_{\Lambda \nu^*} + \nu_{r_i}^*||\hat{y}_{p_i} - \hat{y}_{l_i}||^2_{\Lambda \nu^*} + \nu_{l_i}^*||\hat{y}_{p_i} - \hat{y}_{r_i}||^2_{\Lambda \nu^*}\right)\right)}$$

$$\nu_{p_i}^* = \nu_{p_i} + s_i; \quad \nu_{l_i}^* = \nu_{l_i} + s_{l_i}; \quad \nu_{r_i}^* = \nu_{r_i} + s_{r_i}; \quad \nu^* = \nu_{p_i}^* \nu_{l_i}^* + \nu_{l_i}^* \nu_{r_i}^* + \nu_{r_i}^* \nu_{p_i}^* \qquad (5)$$

where $||.||_\Lambda$ corresponds to the Mahalanobis norm with covariance $\Lambda$. These messages are derived by using the product of Gaussian pdf identities.

## 3.3 MCMC Inference

We propose an MCMC procedure that samples from the posterior distribution over $\pi$ as follows. First, a random node $l$ is pruned from the tree (so that its parent $p_l$ has no parent and only one child), giving the pruned subtree $S_l$ and remaining tree $\pi_l$. See Figure 4. We then consider all possible moves that would place $p_l$ into a valid location elsewhere in the tree. For each branch indexed by the node $i$ "below" it, we compute the posterior density function of where $p_l$ should be placed on that branch. We then slice sample on this collection of density functions. See Figure 5. By cycling through the nodes to prune and reattach, we achieve a Gibbs sampler over $\pi$.

We can efficiently compute the relative change in the likelihood $p(\mathbf{x}|\pi)$ through a combination of belief propagation and local computations. First we perform belief propagation on $\pi_l$ to give upward and downward messages, and on $S_l$ to give only upwards messages. Denote $\pi(S, i, t)$ as the tree formed by attaching $S$ above node $i$ at time $t$ in $\pi$. For the new tree we imagine collecting messages to node $p_l$ resulting in a new factor $Z_{i,l,p_i}(\mathbf{x}, \pi_l(S_l, i, t))$. The messages directly downstream of this factor are $Z_{c(i)}(\mathbf{x}, \pi_l)$ and $Z_{p_{p_i}, r_{p_i}}(\mathbf{x}, \pi_l)$ (if $l_{p_i} = i$, ie $i$ is the "left" child of its parent). If we now imagine that the original likelihood was computed by collecting to node $p_i$, then we see that the first factor should replace the factor $Z_{l_{p_i}, r_{p_i}, p_{p_i}}(\mathbf{x}, \pi_l)$ at node $p_i$ while the latter factor was already included in $Z_{l_{p_i}, r_{p_i}, p_{p_i}}(\mathbf{x}, \pi_l)$. All other factors do no change. The total (multiplicative) change in the likelihood is thus,

$$\Delta Z(\pi_l(S_l, i, t)) = Z_{i,l,p_i}(\mathbf{x}, \pi_l(S_l, i, t)) \frac{Z_{p_{p_i}, r_{p_i}}(\mathbf{x}, \pi_l)}{Z_{l_{p_i}, r_{p_i}, p_{p_i}}(\mathbf{x}, \pi_l)} \tag{6}$$

The update in prior probability for adding the parent of $l$ in the segment $(i, p_i)$ (with times $t_i$ and $t_{p_i}$) at time $t$ is proportional to the product of the Beta pdfs in (2) that arise when $\pi_l(S_l, i, t)$ is constructed, and inversely proportional to the Beta pdf that is removed from $\pi_l$, as well as being proportional to the overall prior probability over $\psi_n$:[4]

$$p(\pi_l(S_l, i, t)) \propto \frac{1}{\mathcal{B}(1 - \frac{t_i}{t_{p_i}}; \alpha, \beta)} \mathcal{B}(1 - \frac{t}{t_{p_i}}; \alpha, \beta) \mathcal{B}(1 - \frac{t_i}{t}; \alpha, \beta) \mathcal{B}(1 - \frac{t_l}{t}; \alpha, \beta) p(\psi(\pi_l(S_l, i, t)))$$
$$\tag{7}$$

Where $\psi(\pi)$ gives the structure part of $\pi = (\psi, \{B_i\})$. $p(\psi(\pi_l(S_l, i, t))$ can be computed for all $i$ in linear time via dynamic programming (it does not depend on the actual value of $t$). By taking the product of (6) and (7) we get the joint posterior of $(i, t)$:

$$p(\pi_l(S_l, i, t)|X) \propto \Delta Z(\pi_l(S_l, i, t)) \, p(\pi_l(S_l, i, t)) \tag{8}$$

$p(\pi_l(S_l, i, t)|X)$ defines the distribution from which we would like to sample. We propose a slice sampler that can propose adding $S_l$ to any segment in $\pi_l$. For a fixed $i$, $p(\pi_l(S_l, i, t)|X)$ is typically unimodal, and typically has a small number of modes at most. If we can find all of the extrema of the posterior, we can easily find the intervals that contain positive probability density for slice sampling moves (see Figure 5). Thus this slice sampling procedure will mix as quickly as slice sampling on a single unimodal distribution. We find the extrema of these functions using Newton methods.

The overall sampling procedure is then to sample a new location for each node (both leaves and internal nodes) of the tree using the Gibbs sampling scheme explained above.

## 3.4 Hyperparameter Inference

As we do not know the structure of the data beforehand, we may not want to predetermine the specific values of $\alpha, \beta$ and $\Lambda$. Thus we define hyperpriors on these parameters and infer them as well. For simplicity we assume the form $\Lambda = kI$ for the Brownian motion covariance parameter. We use an Inverse-Gamma prior on $k$, so that $k^{-1} \sim \mathcal{G}(\kappa, \theta)$.

$$k^{-1}|X \sim \mathcal{G}\left(\frac{(N-1)p}{2} + \kappa, \frac{1}{2}\sum_{i=1}^{N-1} d_i + \theta\right)$$

where $N$ is the number of datapoints, $p$ is the dimension, and $d_i = \frac{||\hat{y}_{l_i} - \hat{y}_{r_i}||^2}{\nu_{l_i} + \nu_{r_i} + s_{l_i} + s_{r_i}}$, $||.||$ is the Euclidean norm.

By putting a $\mathcal{G}(\xi, \lambda)$ prior on $\alpha - 1$ and $\beta - 1$, we achieve a posterior for these parameters:

$$p(\alpha, \beta | \xi, \lambda, X) \propto (\alpha - 1)^\xi (\beta - 1)^\xi e^{-\lambda(\alpha - 1 + \beta - 1)} \prod_{i=1}^{N-1} \frac{1}{B(\alpha, \beta)} \left(1 - \frac{t_i}{t_{p_i}}\right)^{\alpha - 1} \left(\frac{t_i}{t_{p_i}}\right)^{\beta - 1}$$

This posterior is log-concave and thus unimodal. We perform slice sampling to update $\alpha$ and $\beta$.

### 3.5 Predictive Density

Given a set of samples from the posterior, we can approximate the posterior density estimate by sampling a test point located at $y_t$ into each of these trees repeatedly (giving new trees $\pi' = \pi(y_t, i, s)$ for various values of $i$ and $s$) and approximating $p(y_t|X)$ as:

$$p(y_t|X) = \int p(\pi|X)p(y_t|\pi, X)d\pi = \int d\pi p(\pi|X) \int d\pi' p(\pi', y_t|\pi, X)$$

$$\approx \sum_{\pi_i \sim \pi|X} \sum_{\pi'_j \sim p(\pi'_j|\pi_i, X)} p(y_t|\pi'_j, X)$$

where $p(\pi'|\pi, X) = \int p(\pi', y_t|\pi, X)dy_t$. By integrating out $y_{l_i}$ in (5), we get a modification of (8) that is proportional to $p(\pi'|\pi, X)$. Slice sampling from this gives us several new trees $\pi'_j$ for each $\pi_i$, where one of the leaves is not observed. $p(y_t|\pi'_j, X)$ is then available by message passing to the leaf associated with $y_t$ (denoted $l$), which results in a Gaussian over $y_t$. Thus the final approximated density is a mixture of Gaussians with a component for each of the $\pi'_j$.

Performing the aforementioned integration (after replacing $\hat{y}_{l_i}$ with $y_t$), we get:

$$Z_{pred}(i, t) \propto \int dy_t Z_{i, p_i, l}(\mathbf{x}, \pi'(y_t))$$

$$= |2\pi\Lambda|^{-\frac{1}{2}} (\nu_i^* + \nu_{p_i}^*)^{-\frac{k}{2}} \exp\left(-\frac{1}{2}(\nu_l^* + (\nu_i^{*-1} + \nu_{p_i}^{*-1})^{-1})d_{i, p_i}\right)$$

where $d_{i, p_i} = ||\hat{y}_i - \hat{y}_{p_i}||_{\Lambda\nu^*}^2$. This gives the posterior density for the location of the unobserved point:

$$p(\pi' = \pi(\{l\}, i, t)|\pi, X) \propto Z_{pred}(i, t) \frac{Z_{l_i, r_i}(\mathbf{x}, \pi_l)}{Z_{l_i, r_i, p_i}(\mathbf{x}, \pi_l)} p(\pi(\{l\}, i, t))$$

where $p(\pi(\{l\}, i, t))$ is as in (7).

## 4 Experiments

We compare our model to Dirichlet Diffusion Trees (DDT) [16] and to Dirichlet Process Mixtures (DPM) [5, 13]. We used Radford Neal's Flexible Bayesian Modeling package [15] for both the DDT and the DPM experiments. All algorithms were run with vague hyperpriors, except for the DPM concentration parameter which we set to .1 as we did not expect many modes for these experiments.

### 4.1 Synthetic Data

To qualitatively compare our method to Dirichlet Process Mixtures and Dirichlet Diffusion Trees, we ran all three methods on a simulated dataset with $N = 200$, $p = 2$. The data is generated from a mixture of heavy tailed distributions to demonstrate the differences between these algorithms when presented with outliers. As can be seen in Figure 6, the DDT fits a density with reasonably heavy tails, whereas our model fits a narrower distribution. This is a result of the fact that the divergence function of the DDT strongly encourages the branch lengths to be small, and thus a larger variance is required to explain the data. Our model can be combined with a heavy-tailed observation model to produce densities with heavier tails – see the rightmost panel of Figure 6.

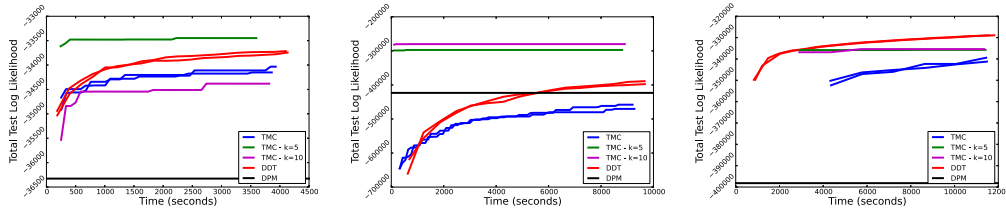

Figure 8: A comparison of our method to the DDT and DPM, using predictive log likelihood on test data. Plots show performance over time, except the DPM which shows the result after convergence. **(left)** the comparison on a $p = 200$ version of the St. Jude's Leukemia dataset. The "TMC - $k$" runs are with $k$ fixed throughout the run. **(middle)** the comparison on the $p = 1000$ version of the Leukemia dataset **(right)** comparison on a $N = 1400, p = 200$ bag of visual words dataset.

## 4.2 Gene Clustering

We applied our model to the St. Jude's Leukemia dataset [22], which has $N_{train} = 215$ datapoints, $N_{test} = 112$, and preprocessed[5] to have $p = 1000$ dimensions. We preprocessed the data so that each dimension had unit variance. Associated with each datapoint is one of 6 classifications of leukemia, or a $7^{th}$ class with which no diagnosis was attributed. We applied our method to the full dataset to see if it could recover these classes. Figure 7 shows the posterior tree sampled after about 28 Gibbs passes (about 10 minutes). We also compared our method against the DDT and DPM on these models' abilities to predict test data on a $p = 200$ subset of the $p = 1000$ dataset, as well as on the $p = 1000$ datset, see Figure 8. On the $p = 200$ dataset, both the DDT and the TMC outperform the DPM, with the TMC performing slightly worse. We attribute this difference in performance due our model's weaker prior on the branch lengths, which causes our model to overfit slightly; if we preset the diffusion variance of our model to a value somewhat larger than the data variance, our performance improves. In the $p = 1000$ dataset, the same phenomenon is observed.

## 4.3 Computer Vision Features

We also cluster visual bag of words features collected from birds images from Visipedia [21]. We worked on a dataset of size $N = 1400$, $N_{test} = 1412$, where each observation belongs to one of 200 classes of birds, see Figure 8. Again our method is better than DPM yet not as well as the DDT. Fixing the variance does improve the performance of our algorithm but not enough to improve over the DDT.

## 5 Conclusion

We introduced a new prior for use in NPB hierarchical clustering, one that can be used in a variety of ways to define a generative model for data. By marginalizing out the time of the coalescent, we achieve a prior from which data can be either generated directly via a graphical model living on trees, or by a CTMC lying on a distribution over times for the branch lengths – in the style of the coalescent and DDT. However, unlike the coalescent and DDT, in our model the times are generated conditioned on the tree structure; giving potential for more interesting models or more efficient inference. The simplicity of the prior allows for efficient Gibbs style inference, and we provide an example model and demonstrate that it can achieve similar performance to that of the DDT. However, to achieve that performance the diffusion variance must be set in advance, suggesting that alternative distributions over the branch lengths may provide better performance than the one explored here.

**Acknowledgements**

This material is based upon work supported by the National Science Foundation under Grant No. 0914783, 0928427, 1018433, 1216045.

## Footnotes

[1]The sequential sampling scheme often associated with NPB models; for example the conditional prior for the CRP is the prior probability of adding the $n+1$st point to one of the existing clusters (or a new cluster) given the clustering on the first $n$ points.

[2]When times are given, we index the internal nodes from most recent to root. Otherwise, nodes are ordered such that parents always succeed children.

[3]It has been brought to our attention that this prior and its connection to the coalescent has been studied before in [2] as the beta-splitting model with parameter $\beta = 0$, and later in [14] under the framework of Gibbs-Fragmentation trees.

[4]Note that if either $l$ or $i$ is a leaf, then the prior term will be simpler than the one listed here

[5]We simply took the 1000 dimensions with the highest variance.

# References

[1] R.P. Adams, Z. Ghahramani, and M.I. Jordan. Tree-structured stick breaking for hierarchical data. *Advances in Neural Information Processing Systems*, 23:19–27, 2010.

[2] D Aldous. Probability distributions on cladograms. *IMA Volumes in Mathematics and its . . .* , 1995.

[3] David Blei, Thomas L. Griffiths, Michael I. Jordan, and Joshua B. Tenenbaum. Hierarchical topic models and the nested chinese restaurant process. In Sebastian Thrun, Lawrence Saul, and Bernhard Schölkopf, editors, *Advances in Neural Information Processing Systems*, volume 16. MIT Press, Cambridge, MA, 2004.

[4] A. Drummond and A. Rambaut. Beast: Bayesian evolutionary analysis by sampling trees. *BMC evolutionary biology*, 7(1):214, 2007.

[5] T.S. Ferguson. Bayesian density estimation by mixtures of normal distributions. *Recent advances in statistics*, pages 287–303, 1983.

[6] D. Görür, L. Boyles, and M. Welling. Scalable inference on kingmanž2019s coalescent using pair similarity. In *Proceedings of AISTATS*, 2012.

[7] D. Görür and Y. W. Teh. An efficient sequential Monte Carlo algorithm for coalescent clustering. In D. Koller, D. Schuurmans, Y. Bengio, and L. Bottou, editors, *Advances in Neural Information Processing Systems 21*, pages 521–528, 2009.

[8] Katherine Heller and Zoubin Ghahramani. Bayesian hierarchical clustering. In *Proceedings of ICML*, volume 22, 2005.

[9] J. F. C. Kingman. The coalescent. *Stochastic Processes and their Applications*, 13:235–248, 1982.

[10] J. F. C. Kingman. On the genealogy of large populations. *Journal of Applied Probability*, 19:27–43, 1982.

[11] D.A. Knowles and Z. Ghahramani. Pitman-yor diffusion trees. *Arxiv preprint arXiv:1106.2494*, 2011.

[12] D.A. Knowles, J. Van Gael, and Z. Ghahramani. Message passing algorithms for dirichlet diffusion trees. In *Proceedings of the 28th Annual International Conference on Machine Learning*, 2011.

[13] A.Y. Lo. On a class of bayesian nonparametric estimates: I. density estimates. *The Annals of Statistics*, 12(1):351–357, 1984.

[14] Peter McCullagh, Jim Pitman, and Matthias Winkel. Gibbs fragmentation trees. *Bernoulli*, 14(4):988–1002, November 2008.

[15] R. Neal. Software for flexible bayesian modeling and markov chain sampling. *see http://www. cs. toronto. edu/ radford/fbm. software. html*, 2003.

[16] R.M. Neal. Density modeling and clustering using dirichlet diffusion trees. *Bayesian Statistics*, 7:619–629, 2003.

[17] A. Rodriguez, D.B. Dunson, and A.E. Gelfand. The nested dirichlet process. *Journal of the American Statistical Association*, 103(483):1131–1154, 2008.

[18] R. Salakhutdinov, J. Tenenbaum, and A. Torralba. Learning to learn with compound hd models. In *Advances in Neural Information Processing Systems 21*, 2012.

[19] J. Steinhardt and Z. Ghahramani. Flexible martingale priors for deep hierarchies. In *International Conference on Artificial Intelligence and Statistics (AISTATS)*, volume 43, pages 61–62, 2012.

[20] Y. W. Teh, H. Daumé III, and D. M. Roy. Bayesian agglomerative clustering with coalescents. In *Advances in Neural Information Processing Systems*, volume 20, 2008.

[21] P. Welinder, S. Branson, T. Mita, C. Wah, F. Schroff, S. Belongie, and P. Perona. Caltech-UCSD Birds 200. Technical Report CNS-TR-2010-001, California Institute of Technology, 2010.

[22] E.J. Yeoh, M.E. Ross, S.A. Shurtleff, W.K. Williams, D. Patel, R. Mahfouz, F.G. Behm, S.C. Raimondi, M.V. Relling, A. Patel, et al. Classification, subtype discovery, and prediction of outcome in pediatric acute lymphoblastic leukemia by gene expression profiling. *Cancer cell*, 1(2):133–143, 2002.

